# Bayesian Policy Gradient Algorithms

**Mohammad Ghavamzadeh**      **Yaakov Engel**
Department of Computing Science, University of Alberta
Edmonton, Alberta, Canada T6E 4Y8
{mgh,yaki}@cs.ualberta.ca

## Abstract

Policy gradient methods are reinforcement learning algorithms that adapt a parameterized policy by following a performance gradient estimate. Conventional policy gradient methods use Monte-Carlo techniques to estimate this gradient. Since Monte Carlo methods tend to have high variance, a large number of samples is required, resulting in slow convergence. In this paper, we propose a Bayesian framework that models the policy gradient as a Gaussian process. This reduces the number of samples needed to obtain accurate gradient estimates. Moreover, estimates of the natural gradient as well as a measure of the uncertainty in the gradient estimates are provided at little extra cost.

## 1   Introduction

Policy Gradient (PG) methods are Reinforcement Learning (RL) algorithms that maintain a parameterized action-selection policy and update the policy parameters by moving them in the direction of an estimate of the gradient of a performance measure. Early examples of PG algorithms are the class of REINFORCE algorithms of Williams [1] which are suitable for solving problems in which the goal is to optimize the average reward. Subsequent work (e.g., [2, 3]) extended these algorithms to the cases of infinite-horizon Markov decision processes (MDPs) and partially observable MDPs (POMDPs), and provided much needed theoretical analysis. However, both the theoretical results and empirical evaluations have highlighted a major shortcoming of these algorithms, namely, the high variance of the gradient estimates. This problem may be traced to the fact that in most interesting cases, the time-average of the observed rewards is a high-variance (although unbiased) estimator of the true average reward, resulting in the sample-inefficiency of these algorithms.

One solution proposed for this problem was to use a small (i.e., smaller than 1) discount factor in these algorithms [2, 3], however, this creates another problem by introducing bias into the gradient estimates. Another solution, which does not involve biasing the gradient estimate, is to subtract a reinforcement baseline from the average reward estimate in the updates of PG algorithms (e.g., [4, 1]). Another approach for speeding-up policy gradient algorithms was recently proposed in [5] and extended in [6, 7]. The idea is to replace the policy-gradient estimate with an estimate of the so-called *natural* policy-gradient. This is motivated by the requirement that a change in the way the policy is parametrized should not influence the result of the policy update. In terms of the policy update rule, the move to a natural-gradient rule amounts to linearly transforming the gradient using the inverse Fisher information matrix of the policy.

However, both conventional and natural policy gradient methods rely on Monte-Carlo (MC) techniques to estimate the gradient of the performance measure. Monte-Carlo estimation is a frequentist procedure, and as such violates the *likelihood principle* [8].[1] Moreover, although MC estimates are unbiased, they tend to produce high variance estimates, or alternatively, require excessive sample sizes (see [9] for a discussion).

In [10] a Bayesian alternative to MC estimation is proposed. The idea is to model integrals of the form $\int f(x)p(x)dx$ as Gaussian Processes (GPs). This is done by treating the first term $f$ in the integrand as a *random function*, the randomness of which reflects our subjective uncertainty concerning its true identity. This allows us to incorporate our prior knowledge on $f$ into its prior distribution. Observing (possibly noisy) samples of $f$ at a set of points $(x_1, x_2, \ldots, x_M)$ allows us to employ Bayes' rule to compute a posterior distribution of $f$, conditioned on these samples. This, in turn, induces a posterior distribution over the value of the integral. In this paper, we propose a Bayesian framework for policy gradient, by modeling the gradient as a GP. This reduces the number of samples needed to obtain accurate gradient estimates. Moreover, estimates of the natural gradient and the gradient covariance are provided at little extra cost.

## 2 Reinforcement Learning and Policy Gradient Methods

Reinforcement Learning (RL) [11, 12] is a class of learning *problems* in which an agent interacts with an unfamiliar, dynamic and stochastic environment, where the agent's goal is to optimize some measure of its long-term performance. This interaction is conventionally modeled as a MDP. Let $\mathcal{P}(\mathcal{S})$ be the set of probability distributions on (Borel) subsets of a set $\mathcal{S}$. A MDP is a tuple $(\mathcal{X}, \mathcal{A}, q, P, P_0)$ where $\mathcal{X}$ and $\mathcal{A}$ are the state and action spaces, respectively; $q(\cdot|a, x) \in \mathcal{P}(\mathbb{R})$ is the probability distribution over rewards; $P(\cdot|a, x) \in \mathcal{P}(\mathcal{X})$ is the transition probability distribution; (we assume that $P$ and $q$ are stationary); and $P_0(\cdot) \in \mathcal{P}(\mathcal{X})$ is the initial state distribution. We denote the random variable distributed according to $q(\cdot|a, x)$ as $r(x, a)$. In addition, we need to specify the rule according to which the agent selects actions at each possible state. We assume that this rule does not depend explicitly on time. A *stationary policy* $\mu(\cdot|x) \in \mathcal{P}(\mathcal{A})$ is a probability distribution over actions, conditioned on the current state. The MDP controlled by the policy $\mu$ induces a Markov chain over state-action pairs. We generically denote by $\xi = (x_0, a_0, x_1, a_1, \ldots, x_{T-1}, a_{T-1}, x_T)$ a path generated by this Markov chain. The probability (or density) of such a path is given by

$$\Pr(\xi|\mu) = P_0(x_0) \prod_{t=0}^{T-1} \mu(a_t|x_t)P(x_{t+1}|x_t, a_t). \tag{1}$$

We denote by $R(\xi) = \sum_{t=0}^{T} \gamma^t r(x_t, a_t)$ the (possibly discounted, $\gamma \in [0, 1]$) *cumulative return* of the path $\xi$. $R(\xi)$ is a random variable both because the path $\xi$ is a random variable, and because even for a given path, each of the rewards sampled in it may be stochastic. The expected value of $R(\xi)$ for a *given* $\xi$ is denoted by $\bar{R}(\xi)$. Finally, let us define the *expected return*,

$$\eta(\mu) = \mathbf{E}(R(\xi)) = \int \bar{R}(\xi) \Pr(\xi|\mu)d\xi. \tag{2}$$

Gradient-based approaches to policy search in RL have recently received much attention as a means to sidetrack problems of partial observability and of policy oscillations and even divergence encountered in value-function based methods (see [11], Sec. 6.4.2 and 6.5.3). In policy gradient (PG) methods, we define a class of smoothly parameterized stochastic policies $\{\mu(\cdot|x; \boldsymbol{\theta}), x \in \mathcal{X}, \boldsymbol{\theta} \in \Theta\}$, estimate the gradient of the expected return (2) with respect to the policy parameters $\boldsymbol{\theta}$ from observed system trajectories, and then improve the policy by adjusting the parameters in the direction of the gradient [1, 2, 3]. The gradient of the expected return $\eta(\boldsymbol{\theta}) = \eta(\mu(\cdot|\cdot; \boldsymbol{\theta}))$ is given by [2]

$$\nabla\eta(\boldsymbol{\theta}) = \int \bar{R}(\xi) \frac{\nabla \Pr(\xi; \boldsymbol{\theta})}{\Pr(\xi; \boldsymbol{\theta})} \Pr(\xi; \boldsymbol{\theta})d\xi, \tag{3}$$

where $\Pr(\xi; \boldsymbol{\theta}) = \Pr(\xi|\mu(\cdot|\cdot; \boldsymbol{\theta}))$. The quantity $\frac{\nabla \Pr(\xi; \boldsymbol{\theta})}{\Pr(\xi; \boldsymbol{\theta})} = \nabla \log \Pr(\xi; \boldsymbol{\theta})$ is known as the *score function* or *likelihood ratio*. Since the initial state distribution $P_0$ and the transition distribution $P$ are independent of the policy parameters $\boldsymbol{\theta}$, we can write the score of a path $\xi$ using Eq. 1 as

$$\boldsymbol{u}(\xi) = \frac{\nabla \Pr(\xi; \boldsymbol{\theta})}{\Pr(\xi; \boldsymbol{\theta})} = \sum_{t=0}^{T-1} \frac{\nabla\mu(a_t|x_t; \boldsymbol{\theta})}{\mu(a_t|x_t; \boldsymbol{\theta})} = \sum_{t=0}^{T-1} \nabla \log \mu(a_t|x_t; \boldsymbol{\theta}). \tag{4}$$

Previous work on policy gradient methods used classical Monte-Carlo to estimate the gradient in Eq. 3. These methods generate i.i.d. sample paths $\xi_1, \ldots, \xi_M$ according to $\Pr(\xi; \boldsymbol{\theta})$, and estimate the gradient $\nabla \eta(\boldsymbol{\theta})$ using the MC estimator

$$\widehat{\nabla \eta}_{MC}(\boldsymbol{\theta}) = \frac{1}{M} \sum_{i=1}^{M} R(\xi_i) \nabla \log \Pr(\xi_i; \boldsymbol{\theta}) = \frac{1}{M} \sum_{i=1}^{M} R(\xi_i) \sum_{t=0}^{T_i - 1} \nabla \log \mu(a_{t,i} | x_{t,i}; \boldsymbol{\theta}). \tag{5}$$

## 3 Bayesian Quadrature

Bayesian quadrature (BQ) [10] is a Bayesian method for evaluating an integral using samples of its integrand. We consider the problem of evaluating the integral

$$\rho = \int f(x) p(x) dx. \tag{6}$$

If $p(x)$ is a probability density function, this becomes the problem of evaluating the expected value of $f(x)$. In MC estimation of such expectations, samples $(x_1, x_2, \ldots, x_M)$ are drawn from $p(x)$, and the integral is estimated as $\hat{\rho}_{MC} = \frac{1}{M} \sum_{i=1}^{M} f(x_i)$. $\hat{\rho}_{MC}$ is an unbiased estimate of $\rho$, with variance that diminishes to zero as $M \to \infty$. However, as O'Hagan points out, MC estimation is fundamentally unsound, as it violates the likelihood principle, and moreover, does not make full use of the data at hand [9].

The alternative proposed in [10] is based on the following reasoning: In the Bayesian approach, $f(\cdot)$ is random simply because it is numerically unknown. We are therefore uncertain about the value of $f(x)$ until we actually evaluate it. In fact, even then, our uncertainty is not always completely removed, since measured samples of $f(x)$ may be corrupted by noise. Modeling $f$ as a Gaussian process (GP) means that our uncertainty is completely accounted for by specifying a Normal prior distribution over functions. This prior distribution is specified by its mean and covariance, and is denoted by $f(\cdot) \sim \mathcal{N}\{f_0(\cdot), k(\cdot, \cdot)\}$. This is shorthand for the statement that $f$ is a GP with prior mean $\mathbf{E}(f(x)) = f_0(x)$ and covariance $\mathbf{Cov}(f(x), f(x')) = k(x, x')$, respectively. The choice of kernel function $k$ allows us to incorporate prior knowledge on the smoothness properties of the integrand into the estimation procedure. When we are provided with a set of samples $\mathcal{D}_M = \{(x_i, y_i)\}_{i=1}^{M}$, where $y_i$ is a (possibly noisy) sample of $f(x_i)$, we apply Bayes' rule to condition the prior on these sampled values. If the measurement noise is normally distributed, the result is a Normal posterior distribution of $f | \mathcal{D}_M$. The expressions for the posterior mean and covariance are standard:

$$\mathbf{E}(f(x) | \mathcal{D}_M) = f_0(x) + \boldsymbol{k}_M(x)^\top \boldsymbol{C}_M (\boldsymbol{y}_M - \boldsymbol{f}_0), \tag{7}$$
$$\mathbf{Cov}(f(x), f(x') | \mathcal{D}_M) = k(x, x') - \boldsymbol{k}_M(x)^\top \boldsymbol{C}_M \boldsymbol{k}_M(x').$$

Here and in the sequel, we make use of the definitions:

$$\boldsymbol{f}_0 = (f_0(x_1), \ldots, f_0(x_M))^\top \quad , \quad \boldsymbol{y}_M = (y_1, \ldots, y_M)^\top,$$
$$\boldsymbol{k}_M(x) = (k(x_1, x), \ldots, k(x_M, x))^\top \quad , \quad [\boldsymbol{K}_M]_{i,j} = k(x_i, x_j) \quad , \quad \boldsymbol{C}_M = (\boldsymbol{K}_M + \boldsymbol{\Sigma}_M)^{-1},$$

and $[\boldsymbol{\Sigma}_M]_{i,j}$ is the measurement noise covariance between the $i$th and $j$th samples. Typically, it is assumed that the measurement noise is i.i.d., in which case $\boldsymbol{\Sigma}_M = \sigma^2 \boldsymbol{I}$, where $\sigma^2$ is the noise variance and $\boldsymbol{I}$ is the identity matrix.

Since integration is a linear operation, the posterior distribution of the integral in Eq. 6 is also Gaussian, and the posterior moments are given by

$$\mathbf{E}(\rho | \mathcal{D}_M) = \int \mathbf{E}(f(x) | \mathcal{D}_M) p(x) dx \quad , \quad \mathbf{Var}(\rho | \mathcal{D}_M) = \iint \mathbf{Cov}(f(x), f(x') | \mathcal{D}_M) p(x) p(x') dx dx'. \tag{8}$$

Substituting Eq. 7 into Eq. 8, we get

$$\mathbf{E}(\rho | \mathcal{D}_M) = \rho_0 + \boldsymbol{z}_M^\top \boldsymbol{C}_M (\boldsymbol{y}_M - \boldsymbol{f}_0) \quad , \quad \mathbf{Var}(\rho | \mathcal{D}_M) = z_0 - \boldsymbol{z}_M^\top \boldsymbol{C}_M \boldsymbol{z}_M, \tag{9}$$

where we made use of the definitions:

$$\rho_0 = \int f_0(x) p(x) dx \quad , \quad \boldsymbol{z}_M = \int \boldsymbol{k}_M(x) p(x) dx \quad , \quad z_0 = \iint k(x, x') p(x) p(x') dx dx'. \tag{10}$$

Note that $\rho_0$ and $z_0$ are the prior mean and variance of $\rho$, respectively.

| | Model 1 | Model 2 |
|---|---|---|
| Known part | $p(\xi;\boldsymbol{\theta}) = \Pr(\xi;\boldsymbol{\theta})$ | $\boldsymbol{p}(\xi;\boldsymbol{\theta}) = \nabla\Pr(\xi;\boldsymbol{\theta})$ |
| Uncertain part | $\boldsymbol{f}(\xi;\boldsymbol{\theta}) = \bar{R}(\xi)\nabla\log\Pr(\xi;\boldsymbol{\theta})$ | $f(\xi) = \bar{R}(\xi)$ |
| Measurement | $\boldsymbol{y}(\xi) = R(\xi)\nabla\log\Pr(\xi;\boldsymbol{\theta})$ | $y(\xi) = R(\xi)$ |
| Prior mean of $f$ | $\mathbf{E}(\boldsymbol{f}(\xi;\boldsymbol{\theta})) = \mathbf{0}$ | $\mathbf{E}(f(\xi)) = 0$ |
| Prior cov. of $f$ | $\mathbf{Cov}(\boldsymbol{f}(\xi;\boldsymbol{\theta}),\boldsymbol{f}(\xi';\boldsymbol{\theta})) = k(\xi,\xi')\boldsymbol{I}$ | $\mathbf{Cov}(f(\xi),f(\xi')) = k(\xi,\xi')$ |
| $\mathbf{E}(\nabla\eta_B(\boldsymbol{\theta})\|\mathcal{D}_M) =$ | $\boldsymbol{Y}_M\boldsymbol{C}_M\boldsymbol{z}_M$ | $\boldsymbol{Z}_M\boldsymbol{C}_M\boldsymbol{y}_M$ |
| $\mathbf{Cov}(\nabla\eta_B(\boldsymbol{\theta})\|\mathcal{D}_M) =$ | $(z_0 - \boldsymbol{z}_M^\top\boldsymbol{C}_M\boldsymbol{z}_M)\boldsymbol{I}$ | $\boldsymbol{Z}_0 - \boldsymbol{Z}_M\boldsymbol{C}_M\boldsymbol{Z}_M^\top$ |
| Kernel function | $k(\xi_i,\xi_j) = \left(1 + \boldsymbol{u}(\xi_i)^\top\boldsymbol{G}^{-1}\boldsymbol{u}(\xi_j)\right)^2$ | $k(\xi_i,\xi_j) = \boldsymbol{u}(\xi_i)^\top\boldsymbol{G}^{-1}\boldsymbol{u}(\xi_j)$ |
| $\boldsymbol{z}_M$ | $(\boldsymbol{z}_M)_i = 1 + \boldsymbol{u}(\xi_i)^\top\boldsymbol{G}^{-1}\boldsymbol{u}(\xi_i)$ | $\boldsymbol{Z}_M = \boldsymbol{U}_M$ |
| $z_0$ | $z_0 = 1 + n$ | $\boldsymbol{Z}_0 = \boldsymbol{G} - \boldsymbol{U}_M\boldsymbol{C}_M\boldsymbol{U}_M^\top$ |

Table 1: Summary of the Bayesian policy gradient Models 1 and 2.

In order to prevent the problem from "degenerating into infinite regress", as phrased by O'Hagan [10], we should choose the functions $p$, $k$, and $f_0$ so as to allow us to solve the integrals in Eq. 10 analytically. For instance, O'Hagan provides the analysis required for the case where the integrands in Eq. 10 are products of multivariate Gaussians and polynomials, referred to as Bayes-Hermite quadrature. One of the contributions of the present paper is in providing analogous analysis for kernel functions that are based on the *Fisher kernel* [13, 14]. It is important to note that in MC estimation, samples must be drawn from the distribution $p(x)$, whereas in the Bayesian approach, samples may be drawn from arbitrary distributions. This affords us with flexibility in the choice of sample points, allowing us, for instance to actively design the samples $(x_1, x_2, \ldots, x_M)$.

## 4 Bayesian Policy Gradient

In this section, we use Bayesian quadrature to estimate the gradient of the expected return with respect to the policy parameters, and propose *Bayesian policy gradient* (BPG) algorithms. In the frequentist approach to policy gradient our performance measure was $\eta(\boldsymbol{\theta})$ from Eq. 2, which is the result of averaging the cumulative return $R(\xi)$ over all possible paths $\xi$ and all possible returns accumulated in each path. In the Bayesian approach we have an additional source of randomness, which is our subjective Bayesian uncertainty concerning the process generating the cumulative returns. Let us denote

$$\eta_B(\boldsymbol{\theta}) = \int R(\xi)\Pr(\xi;\boldsymbol{\theta})d\xi. \tag{11}$$

$\eta_B(\boldsymbol{\theta})$ is a random variable both because of the noise in $R(\xi)$ and the Bayesian uncertainty. Under the quadratic loss, our Bayesian performance measure is $\mathbf{E}(\eta_B(\boldsymbol{\theta})|\mathcal{D}_M)$. Since we are interested in optimizing performance rather than evaluating it, we evaluate the posterior distribution of the gradient of $\eta_B(\boldsymbol{\theta})$. For the mean we have

$$\nabla\mathbf{E}\left(\eta_B(\boldsymbol{\theta})|\mathcal{D}_M\right) = \mathbf{E}\left(\nabla\eta_B(\boldsymbol{\theta})|\mathcal{D}_M\right) = \mathbf{E}\left(\int R(\xi)\frac{\nabla\Pr(\xi;\boldsymbol{\theta})}{\Pr(\xi;\boldsymbol{\theta})}\Pr(\xi;\boldsymbol{\theta})d\xi \,|\mathcal{D}_M\right). \tag{12}$$

Consequently, in BPG we cast the problem of estimating the gradient of the expected return in the form of Eq. 6. As described in Sec. 3, we partition the integrand into two parts, $f(\xi;\boldsymbol{\theta})$ and $p(\xi;\boldsymbol{\theta})$. We will place the GP prior over $f$ and assume that $p$ is known. We will then proceed by calculating the posterior moments of the gradient $\nabla\eta_B(\boldsymbol{\theta})$ conditioned on the observed data. Next, we investigate two different ways of partitioning the integrand in Eq. 12, resulting in two distinct Bayesian models. Table 1 summarizes the two models we use in this work. Our choice of Fisher-type kernels was motivated by the notion that a good representation should depend on the data generating process (see [13, 14] for a thorough discussion). Our particular choices of linear and quadratic Fisher kernels were guided by the requirement that the posterior moments of the gradient be analytically tractable. In Table 1 we made use of the following definitions: $\boldsymbol{F}_M = (\boldsymbol{f}(\xi_1;\boldsymbol{\theta}),\ldots,\boldsymbol{f}(\xi_M;\boldsymbol{\theta})) \sim \mathcal{N}(\mathbf{0},\boldsymbol{K}_M)$, $\boldsymbol{Y}_M = (\boldsymbol{y}(\xi_1),\ldots,\boldsymbol{y}(\xi_M)) \sim \mathcal{N}(\mathbf{0},\boldsymbol{K}_M + \sigma^2\boldsymbol{I})$, $\boldsymbol{U}_M = \left[\boldsymbol{u}(\xi_1)\,,\,\boldsymbol{u}(\xi_2)\,,\,\ldots\,,\,\boldsymbol{u}(\xi_M)\right]$, $\boldsymbol{Z}_M = \int\nabla\Pr(\xi;\boldsymbol{\theta})\boldsymbol{k}_M(\xi)^\top d\xi$, and $\boldsymbol{Z}_0 = \iint k(\xi,\xi')\nabla\Pr(\xi;\boldsymbol{\theta})\nabla\Pr(\xi';\boldsymbol{\theta})^\top d\xi d\xi'$. Finally, $n$ is the number of policy parameters, and $\boldsymbol{G} = \mathbf{E}\left(\boldsymbol{u}(\xi)\boldsymbol{u}(\xi)^\top\right)$ is the Fisher information matrix.

We can now use Models 1 and 2 to define algorithms for evaluating the gradient of the expected return with respect to the policy parameters. The pseudo-code for these algorithms is shown in Alg. 1. The generic algorithm (for either model) takes a set of policy parameters $\boldsymbol{\theta}$ and a sample size $M$ as input, and returns an estimate of the posterior moments of the gradient of the expected return.

---

**Algorithm 1** : A Bayesian Policy Gradient Evaluation Algorithm

---

1: **BPG_Eval**$(\boldsymbol{\theta}, M)$    // policy parameters $\boldsymbol{\theta} \in \mathbb{R}^n$, sample size $M > 0$ //
2: Set $\boldsymbol{G} = \boldsymbol{G}(\boldsymbol{\theta})$   ,   $\mathcal{D}_0 = \emptyset$
3: **for** $i = 1$ to $M$ **do**
4:    Sample a path $\xi_i$ using the policy $\mu(\boldsymbol{\theta})$
5:    $\mathcal{D}_i = \mathcal{D}_{i-1} \bigcup \{\xi_i\}$
6:    Compute $\boldsymbol{u}(\xi_i) = \sum_{t=0}^{T_i-1} \nabla \log \mu(a_t|s_t; \boldsymbol{\theta})$
7:    $R(\xi_i) = \sum_{t=0}^{T_i-1} r(s_t, a_t)$
8:    Update $\boldsymbol{K}_i$ using $\boldsymbol{K}_{i-1}$ and $\xi_i$
9:    $\boldsymbol{y}(\xi_i) = R(\xi_i)\boldsymbol{u}(\xi_i)$           (Model 1)   or   $y(\xi_i) = R(\xi_i)$        (Model 2)
    $(\boldsymbol{z}_M)_i = 1 + \boldsymbol{u}(\xi_i)^\top \boldsymbol{G}^{-1}\boldsymbol{u}(\xi_i)$   (Model 1)   or   $\boldsymbol{Z}_M(:,i) = \boldsymbol{u}(\xi_i)$   (Model 2)
10: **end for**
11: $\boldsymbol{C}_M = (\boldsymbol{K}_M + \sigma^2\boldsymbol{I})^{-1}$
12: Compute the posterior mean and covariance:
    $\mathbf{E}(\nabla\eta_B(\boldsymbol{\theta})|\mathcal{D}_M) = \boldsymbol{Y}_M\boldsymbol{C}_M\boldsymbol{z}_M$   ,   $\mathbf{Cov}(\nabla\eta_B(\boldsymbol{\theta})|\mathcal{D}_M) = (z_0 - \boldsymbol{z}_M^\top\boldsymbol{C}_M\boldsymbol{z}_M)\boldsymbol{I}$   (Model 1)   or
    $\mathbf{E}(\nabla\eta_B(\boldsymbol{\theta})|\mathcal{D}_M) = \boldsymbol{Z}_M\boldsymbol{C}_M\boldsymbol{y}_M$   ,   $\mathbf{Cov}(\nabla\eta_B(\boldsymbol{\theta})|\mathcal{D}_M) = \boldsymbol{Z}_0 - \boldsymbol{Z}_M\boldsymbol{C}_M\boldsymbol{Z}_M^\top$   (Model 2)
13: **return** $\mathbf{E}(\nabla\eta_B(\boldsymbol{\theta})|\mathcal{D}_M)$   ,   $\mathbf{Cov}(\nabla\eta_B(\boldsymbol{\theta})|\mathcal{D}_M)$

---

The kernel functions used in Models 1 and 2 are both based on the Fisher information matrix $\boldsymbol{G}(\boldsymbol{\theta})$. Consequently, every time we update the policy parameters we need to recompute $\boldsymbol{G}$. In Alg. 1 we assume that $\boldsymbol{G}$ is known, however, in most practical situations this will not be the case. Let us briefly outline two possible approaches for estimating the Fisher information matrix.

**MC Estimation:**  At each step $j$, our BPG algorithm generates $M$ sample paths using the current policy parameters $\boldsymbol{\theta}_j$ in order to estimate the gradient $\nabla\eta_B(\boldsymbol{\theta}_j)$. We can use these generated sample paths to estimate the Fisher information matrix $\boldsymbol{G}(\boldsymbol{\theta}_j)$ by replacing the expectation in $\boldsymbol{G}$ with empirical averaging as $\hat{\boldsymbol{G}}_{MC}(\boldsymbol{\theta}_j) = \frac{1}{\sum_{i=1}^M T_i} \sum_{i=1}^M \sum_{t=0}^{T_i-1} \nabla \log \mu(a_t|x_t; \boldsymbol{\theta}_j) \nabla \log \mu(a_t|x_t; \boldsymbol{\theta}_j)^\top$.

**Model-Based Policy Gradient:**  The Fisher information matrix depends on the probability distribution over paths. This distribution is a product of two factors, one corresponding to the current policy, and the other corresponding to the MDP dynamics $P_0$ and $P$ (see Eq. 1). Thus, if the MDP dynamics are known, the Fisher information matrix can be evaluated off-line. We can model the MDP dynamics using some parameterized model, and estimate the model parameters using maximum likelihood or Bayesian methods. This would be a model-based approach to policy gradient, which would allow us to transfer information between different policies.

Alg. 1 can be made significantly more efficient, both in time and memory, by sparsifying the solution. Such sparsification may be performed incrementally, and helps to numerically stabilize the algorithm when the kernel matrix is singular, or nearly so. Here we use an on-line sparsification method from [15] to selectively add a new observed path to a set of *dictionary* paths $\mathcal{D}_M$, which are used as a basis for approximating the full solution. Lack of space prevents us from discussing this method in further detail (see Chapter 2 in [15] for a thorough discussion).

The Bayesian policy gradient (BPG) algorithm is described in Alg. 2. This algorithm starts with an initial vector of policy parameters $\boldsymbol{\theta}_0$ and updates the parameters in the direction of the posterior mean of the gradient of the expected return, computed by Alg. 1. This is repeated $N$ times, or alternatively, until the gradient estimate is sufficiently close to zero.

---

**Algorithm 2** : A Bayesian Policy Gradient Algorithm

---

1: **BPG**$(\boldsymbol{\theta}_0, \boldsymbol{\alpha}, N, M)$      // initial policy parameters $\boldsymbol{\theta}_0$, learning rates $(\alpha_j)_{j=0}^{N-1}$, number of policy updates $N > 0$, **BPG_Eval** sample size $M > 0$ //
2: **for** $j = 0$ to $N - 1$ **do**
3:    $\Delta\boldsymbol{\theta}_j = \mathbf{E}(\nabla\eta_B(\boldsymbol{\theta}_j)|\mathcal{D}_M)$ from **BPG_Eval**$(\boldsymbol{\theta}_j, M)$
4:    $\boldsymbol{\theta}_{j+1} = \boldsymbol{\theta}_j + \alpha_j\Delta\boldsymbol{\theta}_j$  (regular gradient)   or   $\boldsymbol{\theta}_{j+1} = \boldsymbol{\theta}_j + \alpha_j\boldsymbol{G}^{-1}(\boldsymbol{\theta}_j)\Delta\boldsymbol{\theta}_j$  (natural gradient)
5: **end for**
6: **return** $\boldsymbol{\theta}_N$

---

## 5   Experimental Results

In this section, we compare the BQ and MC gradient estimators in a continuous-action bandit problem and a continuous state and action linear quadratic regulation (LQR) problem. We also evaluate

the performance of the BPG algorithm (Alg. 2) on the LQR problem, and compare it with a standard MC-based policy gradient (MCPG) algorithm.

## 5.1 A Bandit Problem

In this simple example, we compare the BQ and MC estimates of the gradient (for a fixed set of policy parameters) using the same samples. Our simple bandit problem has a single state and $\mathcal{A} = \mathbb{R}$. Thus, each path $\xi_i$ consists of a single action $a_i$. The policy, and therefore also the distribution over paths is given by $a \sim \mathcal{N}(\theta_1 = 0, \theta_2^2 = 1)$. The score function of the path $\xi = a$ and the Fisher information matrix are given by $\boldsymbol{u}(\xi) = [a, a^2 - 1]^\top$ and $\boldsymbol{G} = \text{diag}(1, 2)$, respectively.

Table 2 shows the exact gradient of the expected return and its MC and BQ estimates (using 10 and 100 samples) for two versions of the simple bandit problem corresponding to two different deterministic reward functions $r(a) = a$ and $r(a) = a^2$. The average over $10^4$ runs of the MC and BQ estimates and their standard deviations are reported in Table 2. The true gradient is analytically tractable and is reported as "Exact" in Table 2 for reference.

| | Exact | MC (10) | BQ (10) | MC (100) | BQ (100) |
|---|---|---|---|---|---|
| $r(a) = a$ | $\begin{pmatrix} 1 \\ 0 \end{pmatrix}$ | $\begin{pmatrix} 0.9950 \pm 0.438 \\ -0.0011 \pm 0.977 \end{pmatrix}$ | $\begin{pmatrix} 0.9856 \pm 0.050 \\ 0.0006 \pm 0.060 \end{pmatrix}$ | $\begin{pmatrix} 1.0004 \pm 0.140 \\ 0.0040 \pm 0.317 \end{pmatrix}$ | $\begin{pmatrix} 1.000 \pm 0.000001 \\ 0.000 \pm 0.000004 \end{pmatrix}$ |
| $r(a) = a^2$ | $\begin{pmatrix} 0 \\ 2 \end{pmatrix}$ | $\begin{pmatrix} 0.0136 \pm 1.246 \\ 2.0336 \pm 2.831 \end{pmatrix}$ | $\begin{pmatrix} 0.0010 \pm 0.082 \\ 1.9250 \pm 0.226 \end{pmatrix}$ | $\begin{pmatrix} 0.0051 \pm 0.390 \\ 1.9869 \pm 0.857 \end{pmatrix}$ | $\begin{pmatrix} 0.000 \pm 0.000003 \\ 2.000 \pm 0.000011 \end{pmatrix}$ |

Table 2: The true gradient of the expected return and its MC and BQ estimates for two bandit problems.

As shown in Table 2, the BQ estimate has much lower variance than the MC estimate for both small and large sample sizes. The BQ estimate also has a lower bias than the MC estimate for the large sample size ($M = 100$), and almost the same bias for the small sample size ($M = 10$).

## 5.2 A Linear Quadratic Regulator

In this section, we consider the following linear system in which the goal is to minimize the expected return over 20 steps. Thus, it is an episodic problem with paths of length 20.

**System**
Initial State: $x_0 \sim \mathcal{N}(0.3, 0.001)$
Rewards: $r_t = x_t^2 + 0.1a_t^2$
Transitions: $x_{t+1} = x_t + a_t + n_x$; $n_x \sim \mathcal{N}(0, 0.01)$

**Policy**
Actions: $a_t \sim \mu(\cdot|x_t; \boldsymbol{\theta}) = \mathcal{N}(\lambda x_t, \sigma^2)$
Parameters: $\boldsymbol{\theta} = (\lambda, \sigma)^\top$

We first compare the BQ and MC estimates of the gradient of the expected return for the policy induced by the parameters $\lambda = -0.2$ and $\sigma = 1$. We use several different sample sizes (number of paths used for gradient estimation) $M = 5j$, $j = 1, \dots, 20$ for the BQ and MC estimates. For each sample size, we compute both the MC and BQ estimates $10^4$ times, using the same samples. The true gradient is estimated using MC with $10^7$ sample paths for comparison purposes.

Figure 1 shows the mean squared error (MSE) (first column), and the mean absolute angular error (second column) of the MC and BQ estimates of the gradient for several different sample sizes. The absolute angular error is the absolute value of the angle between the true gradient and the estimated gradient. In this figure, the BQ gradient estimate was calculated using Model 1 without sparsification. With a good choice of sparsification threshold, we can attain almost identical results much faster and more efficiently with sparsification. These results are not shown here due to space limitations. To give an intuition concerning the speed and the efficiency attained by sparsification, we should mention that the dimension of the feature space for the kernel used in Model 1 is 6 (Proposition 9.2 in [14]). Therefore, we deal with a kernel matrix of size 6 with sparsification versus a kernel matrix of size $M = 5j$, $j = 1, \dots, 20$ without sparsification.

We ran another set of experiments, in which we add i.i.d. Gaussian noise to the rewards: $r_t = x_t^2 + 0.1a_t^2 + n_r$; $n_r \sim \mathcal{N}(0, \sigma_r^2 = 0.1)$. In Model 2, we can model this by the measurement noise covariance matrix $\boldsymbol{\Sigma} = T\sigma_r^2 \boldsymbol{I}$, where $T = 20$ is the path length. Since each reward $r_t$ is a Gaussian random variable with variance $\sigma_r^2$, the return $R(\xi) = \sum_{t=0}^{T-1} r_t$ will also be a Gaussian random variable with variance $T\sigma_r^2$. The results are presented in the third and fourth columns of Figure 1. These experiments indicate that the BQ gradient estimate has lower variance than its MC counterpart. In fact, whereas the performance of the MC estimate improves as $\frac{1}{M}$, the performance of the BQ estimate improves at a higher rate.

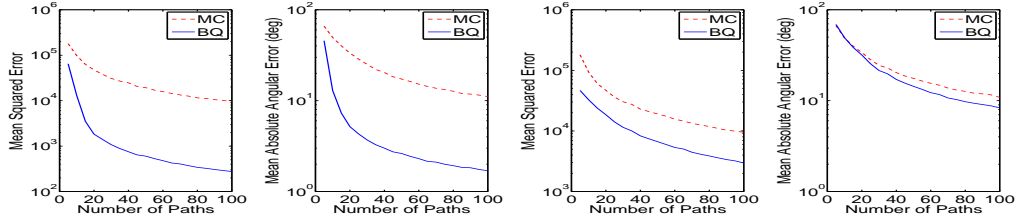

Figure 1: Results for the LQR problem using Model 1 (left) and Model 2 (right), without sparsification. The Model 2 results are for a LQR problem, in which the rewards are corrupted by i.i.d. Gaussian noise. For each algorithm, we show the MSE (left) and the mean absolute angular error (right), as functions of the number of sample paths $M$. Note that the errors are plotted on a logarithmic scale. All results are averages over $10^4$ runs.

Next, we use BPG to optimize the policy parameters in the LQR problem. Figure 2 shows the performance of the BPG algorithm with the regular (BPG) and the natural (BPNG) gradient estimates, versus a MC-based policy gradient (MCPG) algorithm, for the sample sizes (number of sample paths used for estimating the gradient of a policy) $M = 5, 10, 20$, and $40$. We use Alg. 2 with the number of updates set to $N = 100$, and Model 1 for the BPG and BPNG methods. Since Alg. 2 computes the Fisher information matrix for each set of policy parameters, an estimate of the natural gradient is provided at little extra cost at each step. The returns obtained by these methods are averaged over $10^4$ runs for sample sizes $5$ and $10$, and over $10^3$ runs for sample sizes $20$ and $40$. The policy parameters are initialized randomly at each run. In order to ensure that the learned parameters do not exceed an acceptable range, the policy parameters are defined as $\lambda = -1.999 + 1.998/(1 + e^{\nu_1})$ and $\sigma = 0.001 + 1/(1 + e^{\nu_2})$. The optimal solution is $\lambda^* \approx -0.92$ and $\sigma^* = 0.001$ ($\eta_B(\lambda^*, \sigma^*) = 0.1003$) corresponding to $\nu_1^* \approx -0.16$ and $\nu_2^* \to \infty$.

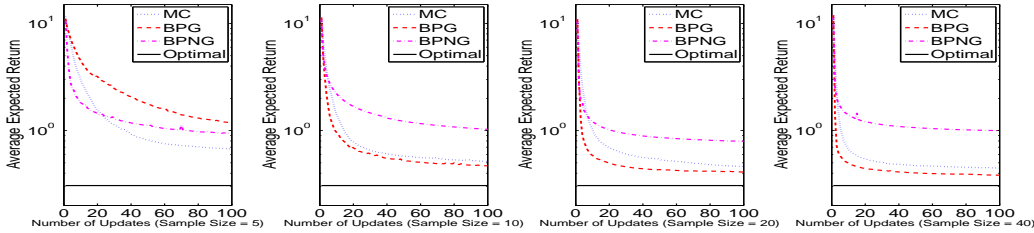

Figure 2: A comparison of the average expected returns of BPG using regular (BPG) and natural (BPNG) gradient estimates, with the average expected return of the MCPG algorithm for sample sizes $5, 10, 20$, and $40$.

Figure 2 shows that MCPG performs better than the BPG algorithm for the smallest sample size ($M = 5$), whereas for larger samples BPG dominates MCPG. This phenomenon is also reported in [16]. We use two different learning rates for the two components of the gradient. For a fixed sample size, each method starts with an initial learning rate, and decreases it according to the schedule $\alpha_j = \alpha_0(20/(20 + j))$. Table 3 summarizes the best initial learning rates for each algorithm. The selected learning rates for BPNG are significantly larger than those for BPG and MCPG, which explains why BPNG initially learns faster than BPG and MCPG, but contrary to our expectations, eventually performs worse.

So far we have assumed that the Fisher information matrix is known. In the next experiment, we estimate it using both MC and maximum likelihood (ML) methods as described in Sec. 4. In ML estimation, we assume that the

|       | $M = 5$     | $M = 10$    | $M = 20$    | $M = 40$    |
|-------|-------------|-------------|-------------|-------------|
| MCPG  | 0.01, 0.05  | 0.05, 0.10  | 0.05, 0.10  | 0.10, 0.15  |
| BPG   | 0.01, 0.03  | 0.07, 0.10  | 0.15, 0.20  | 0.10, 0.30  |
| BPNG  | 0.03, 0.50  | 0.09, 0.30  | 0.45, 0.90  | 0.80, 0.90  |

Figure 3: Initial learning rates used by the PG algorithms.

transition probability function is $P(x_{t+1}|x_t, a_t) = \mathcal{N}(\beta_1 x_t + \beta_2 a_t + \beta_3, \beta_4^2)$, and then estimate its parameters by observing state transitions. Figure 4 shows that when the Fisher information matrix is estimated using MC (BPG-MC), the BPG algorithm still performs better than MCPG, and outperforms the BPG algorithm in which the Fisher information matrix is estimated using ML (BPG-ML). Moreover, as we increase the sample size, its performance converges to the performance of the BPG algorithm in which the Fisher information matrix is known (BPG).

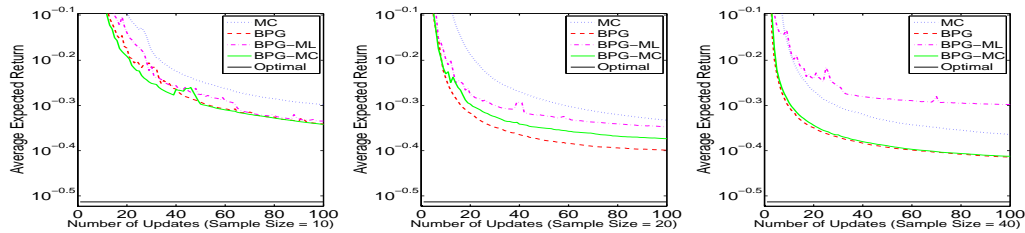

Figure 4: A comparison of the average return of BPG when the Fisher information matrix is known (BPG), and when it is estimated using MC (BPG-MC) and ML (BPG-ML) methods, for sample sizes 10, 20, and 40 (from left to right). The average return of the MCPG algorithm is also provided for comparison.

## 6   Discussion

In this paper we proposed an alternative approach to conventional frequentist policy gradient estimation procedures, which is based on the Bayesian view. Our algorithms use GPs to define a prior distribution over the gradient of the expected return, and compute the posterior, conditioned on the observed data. The experimental results are encouraging, but we conjecture that even higher gains may be attained using this approach. This calls for additional theoretical and empirical work.

Although the proposed policy updating algorithm (Alg. 2) uses only the posterior mean of the gradient in its updates, we hope that more elaborate algorithms can be devised that would make judicious use of the covariance information provided by the gradient estimation algorithm (Alg. 1). Two obvious possibilities are: 1) risk-aware selection of the update step-size and direction, and 2) using the variance in a termination condition for Alg. 1. Other interesting directions include 1) investigating other possible partitions of the integrand in the expression for $\nabla \eta_B(\boldsymbol{\theta})$ into a GP term $f$ and a known term $p$, 2) using other types of kernel functions, such as sequence kernels, 3) combining our approach with MDP model estimation, to allow transfer of learning between different policies, 4) investigating methods for learning the Fisher information matrix, 5) extending the Bayesian approach to Actor-Critic type of algorithms, possibly by combining BPG with the Gaussian process temporal difference (GPTD) algorithms of [15].

**Acknowledgments**    We thank Rich Sutton and Dale Schuurmans for helpful discussions. M.G. would like to thank Shie Mannor for his useful comments at the early stages of this work. M.G. is supported by iCORE and Y.E. is partially supported by an Alberta Ingenuity fellowship.

## Footnotes

[1]The likelihood principle states that in a parametric statistical model, all the information about a data sample that is required for inferring the model parameters is contained in the likelihood function of that sample.

[2]Throughout the paper, we use the notation $\nabla$ to denote $\nabla_{\boldsymbol{\theta}}$ – the gradient w.r.t. the policy parameters.

## References

[1] R. Williams. Simple statistical gradient-following algorithms for connectionist reinforcement learning. *Machine Learning*, 8:229–256, 1992.

[2] P. Marbach. *Simulated-Based Methods for Markov Decision Processes*. PhD thesis, MIT, 1998.

[3] J. Baxter and P. Bartlett. Infinite-horizon policy-gradient estimation. *JAIR*, 15:319–350, 2001.

[4] R. Sutton, D. McAllester, S. Singh, and Y. Mansour. Policy gradient methods for reinforcement learning with function approximation. In *Proceedings of NIPS 12*, pages 1057–1063, 2000.

[5] S. Kakade. A natural policy gradient. In *Proceedings of NIPS 14*, 2002.

[6] J. Bagnell and J. Schneider. Covariant policy search. In *Proceedings of the 18th IJCAI*, 2003.

[7] J. Peters, S. Vijayakumar, and S. Schaal. Reinforcement learning for humanoid robotics. In *Proceedings of the Third IEEE-RAS International Conference on Humanoid Robots*, 2003.

[8] J. Berger and R. Wolpert. *The Likelihood Principle*. Inst. of Mathematical Statistics, Hayward, CA, 1984.

[9] A. O'Hagan. Monte Carlo is fundamentally unsound. *The Statistician*, 36:247–249, 1987.

[10] A. O'Hagan. Bayes-Hermite quadrature. *Journal of Statistical Planning and Inference*, 29, 1991.

[11] D. Bertsekas and J. Tsitsiklis. *Neuro-Dynamic Programming*. Athena Scientific, 1996.

[12] R. Sutton and A. Barto. *An Introduction to Reinforcement Learning*. MIT Press, 1998.

[13] T. Jaakkola and D. Haussler. Exploiting generative models in discriminative classifiers. In *Proceedings of NIPS 11*. MIT Press, 1998.

[14] J. Shawe-Taylor and N. Cristianini. *Kernel Methods for Pattern Analysis*. Cambridge Univ. Press, 2004.

[15] Y. Engel. *Algorithms and Representations for Reinforcement Learning*. PhD thesis, The Hebrew University of Jerusalem, Israel, 2005.

[16] C. Rasmussen and Z. Ghahramani. Bayesian Monte Carlo. In *Proceedings of NIPS 15*. MIT Press, 2003.